# Correlation Functions in a Large Stochastic Neural Network

**Iris Ginzburg**
School of Physics and Astronomy
Raymond and Beverly Sackler Faculty of Exact Sciences
Tel-Aviv University
Tel-Aviv 69978, Israel

**Haim Sompolinsky**
Racah Institute of Physics and Center for Neural Computation
Hebrew University
Jerusalem 91904, Israel

## Abstract

Most theoretical investigations of large recurrent networks focus on the properties of the macroscopic order parameters such as population averaged activities or average overlaps with memories. However, the statistics of the fluctuations in the local activities may be an important testing ground for comparison between models and observed cortical dynamics. We evaluated the neuronal correlation functions in a stochastic network comprising of excitatory and inhibitory populations. We show that when the network is in a stationary state, the cross-correlations are relatively weak, i.e., their amplitude relative to that of the auto-correlations are of order of $1/N$, $N$ being the size of the interacting population. This holds except in the neighborhoods of bifurcations to nonstationary states. As a bifurcation point is approached the amplitude of the cross-correlations grows and becomes of order 1 and the decay time-constant diverges. This behavior is analogous to the phenomenon of *critical slowing down* in systems at thermal equilibrium near a critical point. Near a Hopf bifurcation the cross-correlations exhibit damped oscillations.

# 1  INTRODUCTION

In recent years there has been a growing interest in the study of cross-correlations between the activities of pairs of neurons in the cortex. In many cases the cross-correlations between the activities of cortical neurons are approximately symmetric about zero time delay. These have been taken as an indication of the presence of "functional connectivity" between the correlated neurons (Fetz, Toyama and Smith 1991, Abeles 1991). However, a quantitative comparison between the observed cross-correlations and those expected to exist between neurons that are part of a large assembly of interacting population has been lacking.

Most of the theoretical studies of recurrent neural network models consider only time averaged firing rates, which are usually given as solutions of mean-field equations. They do not account for the fluctuations about these averages, the study of which requires going beyond the mean-field approximations. In this work we perform a theoretical study of the fluctuations in the neuronal activities and their correlations, in a large stochastic network of excitatory and inhibitory neurons. Depending on the model parameters, this system can exhibit coherent undamped oscillations. Here we focus on parameter regimes where the system is in a statistically stationary state, which is more appropriate for modeling non oscillatory neuronal activity in cortex. Our results for the magnitudes and the time-dependence of the correlation functions can provide a basis for comparison with physiological data on neuronal correlation functions.

# 2  THE NEURAL NETWORK MODEL

We study the correlations in the activities of neurons in a fully connected recurrent network consisting of excitatory and inhibitory populations. The excitatory connections between all pairs of excitatory neurons are assumed to be equal to $J/N$ where $N$ denotes the number of excitatory neurons in the network. The excitatory connections from each of the excitatory neurons to each of the inhibitory neurons are $J'/N$. The inhibitory coupling of each of the inhibitory neurons onto each of the excitatory neurons is $K/M$ where $M$ denotes the number of inhibitory neurons. Finally, the inhibitory connections between pairs of inhibitory neurons are $K'/M$. The values of these parameters are in units of the amplitude of the local noise (see below). Each neuron has two possible states, denoted by $s_i = \pm 1$ and $\sigma_i = \pm 1$ for the $i$-th excitatory and inhibitory neurons, respectively. The value $-1$ denotes a quiet state. The value $+1$ denotes an active state that corresponds to a state with high firing rate. The neurons are assumed to be exposed to local noise resulting in stochastic dynamics of their states. This dynamics is specified by transition probabilities between the $-1$ and $+1$ states that are sigmoidal functions of their local fields. The local fields of the $i$-th excitatory neuron, $E_i$ and the $i$-th inhibitory neuron, $I_i$, at time $t$, are

$$E_i(t) = Js(t) - K\sigma(t) - \theta \tag{1}$$

$$I_i(t) = J's(t) - K'\sigma(t) - \theta \tag{2}$$

where $\theta$ represents the local threshold and $s$ and $\sigma$ are the population-averaged activities $s(t) = 1/N \sum_j s_j(t)$, and $\sigma(t) = 1/M \sum_j \sigma_j(t)$ of the excitatory and inhibitory neurons, respectively.

# 3    AVERAGE FIRING RATES

The macroscopic state of the network is characterized by the dynamics of $s(t)$ and $\sigma(t)$. To leading order in $1/N$ and $1/M$, they obey the following well known equations

$$\tau_0 \frac{ds}{dt} = -s + \tanh\left(Js - K\sigma - \theta\right) \tag{3}$$

$$\tau_0 \frac{d\sigma}{dt} = -\sigma + \tanh\left(J's - K'\sigma - \theta\right) \tag{4}$$

where $\tau_0$ is the microscopic time constant of the system. Equations of this form for the two population dynamics have been studied extensively by Wilson and Cowan (Wilson and Cowan 1972) and others (Schuster and Wagner 1990, Grannan, Kleinfeld and Sompolinsky 1992)

Depending on the various parameters the stable solutions of these equations are either fixed-points or limit cycles. The fixed-point solutions represent a stationary state of the network in which the population-averaged activities are almost constant in time. The limit-cycle solutions represent nonstationary states in which there is a coherent oscillatory activity. Obviously in the latter case there are strong oscillatory correlations among the neurons. Here we focus on the fixed-point case. It is described by the following equations

$$s_0 = \tanh\left(Js_0 - K\sigma_0 - \theta\right) \tag{5}$$

$$\sigma_0 = \tanh\left(J's_0 - K'\sigma_0 - \theta\right) \tag{6}$$

where $s_0$ and $\sigma_0$ are the fixed-point values of $s$ and $\sigma$. Our aim is to estimate the magnitude of the correlations between the temporal fluctuations in the activities of neurons in this statistically stationary state.

# 4    CORRELATION FUNCTIONS

There are two types of auto-correlation functions, for the two different populations. For the excitatory neurons we define the auto-correlations as:

$$C_{ii}(\tau) \equiv \langle \delta s_i(t)\delta s_i(t + \tau)\rangle_t \tag{7}$$

where $\delta s_i(t) = s_i(t) - s_0$ and $< ... >_t$ means average over time $t$. A similar definition holds for the auto-correlations of the inhibitory neurons. In our network there are three different cross-correlations: excitatory-excitatory, inhibitory- inhibitory, and inhibitory-excitatory. The excitatory-excitatory correlations are

$$C_{ij}(\tau) \equiv \langle \delta s_i(t)\delta s_j(t + \tau)\rangle_t \ . \tag{8}$$

Similar definitions hold for the other functions.

We have evaluated these correlation functions by solving the equations for the correlations of $\delta s_i(t)$ in the limit of large $N$ and $M$. We find the following forms for the correlations:

$$C_{ii}(\tau) \approx (1 - s_0^2)\exp(-\lambda_1\tau) + \frac{1}{N}\sum_{l=1}^{3} a_l \exp(-\lambda_l\tau) \tag{9}$$

$$C_{ij}(\tau) \approx \frac{1}{N}\sum_{l=1}^{3} b_l \exp(-\lambda_l\tau) \ . \tag{10}$$

The coefficients $a_l$ and $b_l$ are in general of order 1. The three $\lambda_l$ represent three inverse time-constants in our system, where $Re(\lambda_1) \geq Re(\lambda_2) \geq Re(\lambda_3)$. The first inverse time constant equals simply to $\lambda_1 = 1/\tau_0$, and corresponds to a purely local mode of fluctuations. The values of $\lambda_2$ and $\lambda_3$ depend on the parameters of the system. They represent two collective modes of fluctuations that are coherent across the populations. An important outcome of our analysis is that $\lambda_2$ and $\lambda_3$ are exactly the eigenvalues of the stability matrix obtained by linearizing Eqs. (3) and (4)  about the fixed-point Eqs. (5)  and (6) .

The above equations imply two differences between the auto-correlations and the cross-correlations. First, $C_{ii}$ are of order 1 whereas in general $C_{ij}$ is of $O(1/N)$. Secondly, the time-dependence of $C_{ii}$ is dominated by the local, fast time constant $\tau_0$, whereas $C_{ij}$ may be dominated by the slower, collective time-constants.

The conclusion that the cross-correlations are small relative to the auto-correlations might break down if the coefficients $b_l$ take anomalously large values. To check these possibility we have studied in detail the behavior of the correlations near bifurcation points, at which the fixed point solutions become unstable. For concreteness we will discuss here the case of Hopf bifurcations. (Similar results hold for other bifurcations as well). Near a Hopf bifurcation $\lambda_2$ and $\lambda_3$ can be written as $\lambda_\pm \approx \epsilon \pm i\omega$, where $\epsilon > 0$ and vanishes at the bifurcation point. In this parameter regime, the amplitudes $b_1 << b_2, b_3$ and $b_2 \approx b_3 \approx \frac{1}{\epsilon}$. Similar results hold for $a_2$ and $a_3$. Thus, near the bifurcation, we have

$$C_{ii}(\tau) \approx (1 - s_0^2)\exp(-\tau/\tau_0)cos(\omega\tau) \tag{11}$$

$$C_{ij}(\tau) \approx \frac{B}{N\epsilon}\exp(-\epsilon\tau)cos(\omega\tau) \ . \tag{12}$$

Note that near a bifurcation point $\epsilon$ is linear in the difference between any of the parameters and their value at the bifurcation. The above expressions hold for $\epsilon << 1$ but large compared to $1/N$. When $\epsilon \leq 1/N$ the cross-correlation becomes of order 1, and remains so throughout the bifurcation.

Figures 1 and 2 summarize the results of Eqs. (9)  and (10)  near the Hopf bifurcation point at $J, J', K, K', \theta = 225, 65, 161, 422, 2.4$. The population sizes are $N = 10000, M = 1000$. We have chosen a parameter range so that the fixed point values of $s_0$ and $\sigma_0$ will represent a state with low firing rate resembling the spontaneous activity levels in the cortex. For the above parameters the rates relative to the saturation rates are 0.01 and 0.03 for the excitatory and inhibitory populations respectively.

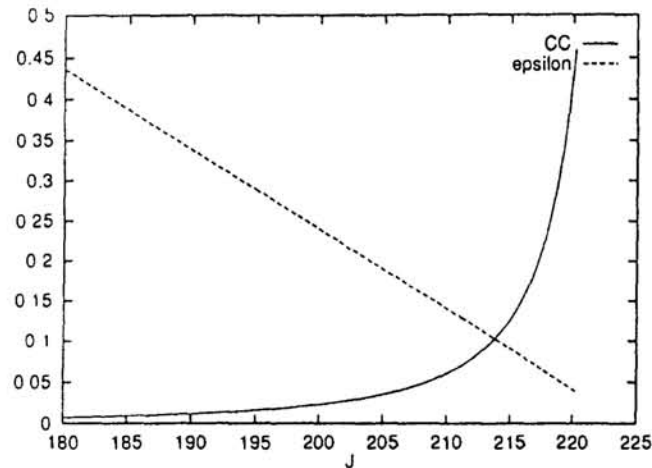

FIGURE 1. The equal-time cross-correlations between a pair of excitatory neurons, and the real part of its inverse time-constant, $\epsilon$, vs. the excitatory coupling parameter J.

The values of $C_{ij}(0)$ and of the real-part of the inverse-time constants of $C_{ij}$ are plotted (Fig. 1) as a function of the parameter $J$ holding the rest of the parameters fixed at their values at the bifurcation point. Thus in this case $\epsilon \ \alpha (225 - J)$. The Figure shows the growth of $C_{ij}$ and the vanishing of the inverse time constant as the bifurcation point is approached.

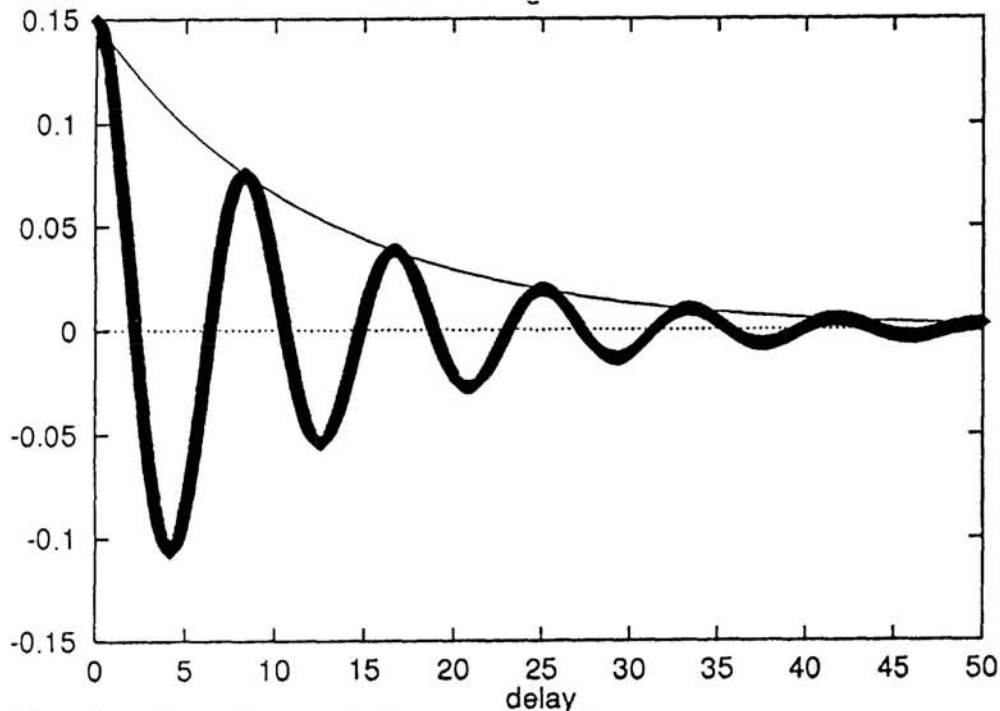

The time-dependence of the cross-correlations near the bifurcation ($J = 215$) is shown in Fig. 2. Time is plotted in units of $\tau_0$. The pronounced damped oscillations are, according to our theory, characteristic of the behavior of the correlations near but below a Hopf bifurcation.

# 5  CONCLUSION

Most theoretical investigations of large recurrent networks focus on the properties of the macroscopic order parameters such as population averaged activity or average overlap with memories. However, the statistics of the fluctuations in the activities may be an important testing ground for comparison between models and observed cortical dynamics. We have studied the properties of the correlation functions in a stochastic network comprising of excitatory and inhibitory populations. We have shown that the cross-correlations are relatively weak in stationary states, except in the neighborhoods of bifurcations to nonstationary states. The growth of the amplitude of these correlations is coupled to a growth in the correlation time-constant. This divergence of the correlation time is analogous to the phenomenon of *critical slowing down* in systems at thermal equilibrium near a critical point. Our analysis can be extended to stochastic networks consisting of a small number of interacting homogeneous populations.

Detailed comparison between the model's results and experimental values of auto- and cross- correlograms of extracellularly measured spike trains in the neocortex have been carried out (Abeles, Ginzburg and Sompolinsky). The tentative conclusion of this study is that the magnitude of the observed correlations and their time-dependence are inconsistent with the expected ones for a system in a stationary state. They therefore indicate that cortical neuronal assemblies are in a nonstationary (but aperiodic) dynamic state.

**Acknowledgements:** We thank M. Abeles for most helpful discussions. This work is partially supported by the USA-Israel Binational Science Foundation.

## REFERENCES

Abeles M., 1991. Corticonics: Neural Circuits of the Cerebral Cortex. Cambridge University Press.

Abeles M., Ginzburg I. & Sompolinsky H. Neuronal Cross-Correlations and Organized Dynamics in the Neocortex. to appear

Fetz E., Toyama K. & Smith W., 1991. Synaptic Interactions Between Cortical Neurons. Cerebral Cortex, edited by A. Peters & G. Jones Plenum Press,NY. Vol 9. 1-43.

Grannan E., Kleinfeld D. & Sompolinsky H., 1992. Stimulus Dependent Synchronization of Neuronal Assemblies. Neural Computation 4,550-559.

Schuster H. G. & Wagner P., 1990. Biol. Cybern. 64, 77.

Wilson H. R. & Cowan J. D., 1972. Excitatory and Inhibitory Interactions in Localized Populations of Model Neurons. Biophy. J. 12, 1-23.